# Mutagenetic tree Fisher kernel improves prediction of HIV drug resistance from viral genotype

**Tobias Sing**
Department of Computational Biology
Max Planck Institute for Informatics
Saarbrücken, Germany
`tobias.sing@mpi-sb.mpg.de`

**Niko Beerenwinkel**[*]
Department of Mathematics
University of California
Berkeley, CA 94720

## Abstract

Starting with the work of Jaakkola and Haussler, a variety of approaches have been proposed for coupling domain-specific generative models with statistical learning methods. The link is established by a kernel function which provides a similarity measure based inherently on the underlying model. In computational biology, the full promise of this framework has rarely ever been exploited, as most kernels are derived from very generic models, such as sequence profiles or hidden Markov models. Here, we introduce the MTreeMix kernel, which is based on a generative model tailored to the underlying biological mechanism. Specifically, the kernel quantifies the similarity of evolutionary escape from antiviral drug pressure between two viral sequence samples. We compare this novel kernel to a standard, evolution-agnostic amino acid encoding in the prediction of HIV drug resistance from genotype, using support vector regression. The results show significant improvements in predictive performance across 17 anti-HIV drugs. Thus, in our study, the generative-discriminative paradigm is key to bridging the gap between population genetic modeling and clinical decision making.

## 1 Introduction

Kernels provide a general framework of statistical learning that allows for integrating problem-specific background knowledge via the geometry of a feature space. Owing to this unifying characteristic, kernel methods enjoy increasing popularity in many application domains, particularly in computational biology [1]. Unfortunately, despite some basic results on the derivation of novel kernels from existing kernels or from more general similarity measures (e.g. via the empirical kernel map [1]), the field suffers from a lack of well-characterized design principles. As a consequence, most novel kernels are still developed in an ad hoc manner.

One of the most promising developments in the recent search for a systematic kernel design methodology is the generative-discriminative paradigm [2], also known under the more general term of model-dependent feature extraction (MDFE) [3]. The central idea of MDFE is to derive kernels from generative probabilistic models of a given process or phenomenon. Starting with Jaakkola and Haussler [2] and the seminal work of Amari [4] on the differential geometric structure of probabilistic models, a number of studies have contributed to an emerging theoretical foundation of MDFE. However, the paradigm is also of immediate intuitive appeal, because mechanistic models of a process that are consistent with observed data and that provide falsifiable predictions often allow for more profound insights than purely discriminative approaches. Moreover, entities that are similar according to a mechanistic model should be expected to exhibit similar behavior in any related prop-

[*]Current address: Program for Evolutionary Dynamics, Harvard University, Cambridge, MA 02138, `beerenw@fas.harvard.edu`

erties. From this perspective, MDFE provides a natural bridge between mathematical modeling and statistical learning.

To date, a variety of generic MDFE procedures have been proposed, including the Fisher kernel [2] and, more generally, marginalized kernels [5], as well as the TOP [3], heat [6], and probability product kernels [7], along with a number of variations. Surprisingly, however, instantiations of these procedures in bioinformatics have been confined to a very limited number of classical problems, namely protein fold recognition, DNA splice site prediction, exon detection, and phylogenetics. Furthermore, most approaches are based on standard graphical models, such as amino acid sequence profiles or hidden Markov models, that are not adapted in any specific way to the process at hand. For example, a first-order Markov chain along the primary structure of a protein is hardly related to the causal mechanisms underlying polypeptide evolution. Thus, the potential of combining biological modeling with kernelization in the framework of MDFE remains vastly unexplored.

This paper is motivated by a regression problem from clinical bioinformatics that has recently attracted substantial attention due to its pivotal role in anti-HIV therapy: the prediction of phenotypic drug resistance from viral genotype (reviewed in [8]). Drug resistant viruses present a major cause of treatment failure and their occurrence renders many of the available drugs ineffective. Therefore, knowing the precise patterns of drug resistance is an important prerequisite for the choice of optimal drug combinations [9, 10].

Drug resistance arises as a virus population evolves under partially suppressive antiviral therapy. The extreme evolutionary dynamics of HIV quickly generate viral genetic variants that are selected for their ability to replicate in the presence of the applied drug cocktail. These advantageous mutants eventually outgrow the wild type population and lead to therapy failure. Thus, the resistance phenotype is determined by the viral genotype. The genotype-phenotype prediction problem is of considerable clinical relevance, because genotyping is much faster and cheaper, while treatment decisions are ultimately based on the viral phenotype (i.e. the level of resistance).

From the perspective of MDFE, the interesting feature of HIV drug resistance lies in the structure of the underlying generative process. The development of resistance involves the stochastic accumulation of mutations in the viral genome along certain mutational pathways. Here, we demonstrate how to exploit this evolutionary structure in genotype-phenotype prediction by deriving a Fisher kernel for mixtures of mutagenetic trees, a family of graphical models designed to represent such genetic accumulation processes. The remainder of this paper is organized as follows. In the next section, we briefly summarize the mutagenetic trees mixture (MTreeMix) model, originally introduced in [11]. The Fisher kernel is derived in Section 3. In Section 4, the kernel is applied to the genotype-phenotype prediction problem introduced above. We conclude with some of the broader implications of our study, including directions for future work.

## 2 Mixture models of mutagenetic trees

Consider $n$ genetic events $\{1, \ldots, n\}$. With each event $v$, we associate the binary random variable $X_v$, such that $\{X_v = 1\}$ indicates the occurrence of $v$. In our applications, the set $\{1, \ldots, n\}$ will denote the mutations conferring resistance to a specific anti-HIV drug. Syntactically, a *mutagenetic tree* for $n$ genetic events is a connected branching $T = (V, E)$ on the vertices $V = \{0, 1, \ldots, n\}$ and rooted at 0, where $E \subseteq V \times V$ denotes the edge set of $T$. Semantically, the *mutagenetic tree model* induced by $T$ and the parameter vector $\theta = (\theta_1, \ldots, \theta_n) \in (0, 1)^n$ is the Bayesian network on $T$ with constrained conditional probability tables of the form

$$\vartheta_v = \begin{array}{c} 0 \\ 1 \end{array} \begin{pmatrix} 1 & 0 \\ 1 - \theta_v & \theta_v \end{pmatrix}, \quad v = 1, \ldots n.$$

Thus, a mutagenetic tree model is the family of distributions of $X = (X_1, \ldots, X_n)$ that factor as

$$\Pr(X = x \mid \theta) = \prod_{v=1}^{n} \vartheta_{v, (x_{\mathrm{pa}(v)}, x_v)}.$$

Here, $x_0 := 1$ (indicating the wild type state without any resistance mutations), and $\mathrm{pa}(v)$ denotes the parent of vertex $v$ in $T$. Figure 1 shows a mutagenetic tree for the development of resistance to the protease inhibitor nelfinavir.

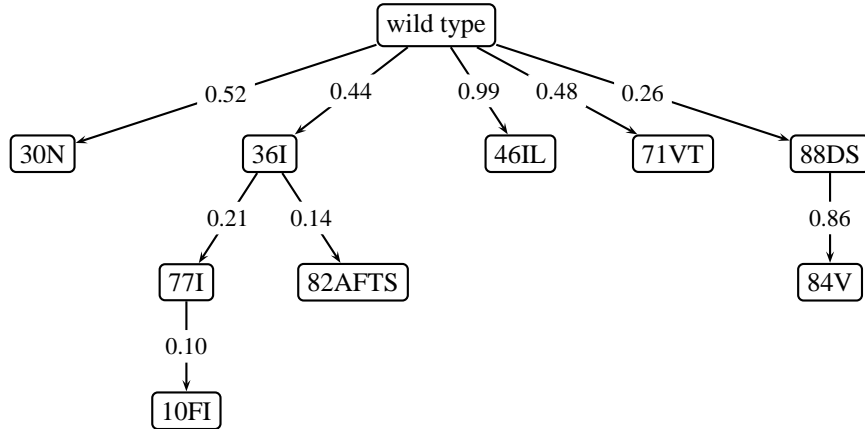

Figure 1: Mutagenetic tree for the development of resistance to the HIV protease inhibitor nelfinavir (NFV). Vertices of the tree are labeled with amino acid changes in the protease enzyme. Edges are labeled with conditional probabilities. The tree represents one component of the 6-trees mixture model estimated for this evolutionary process.

The probability tables impose the constraint that a mutation can only be present if its predecessor in the topology is also present. This restriction sets mutagenetic trees apart from standard Bayesian networks in that it allows for an evolutionary interpretation of the tree topology. In particular, the model implies the existence of certain mutational pathways with distinct probabilities. Each pathway is required to respect the order of mutation accumulation that is encoded in the tree. Mutational patterns which do not respect these order constraints have probability zero in the model. We shall exclude these genotypes from the state space of the model. The state space then becomes the following subset of $\{0, 1\}^n$,

$$\mathcal{C} = \{x \in \{0, 1\}^n \mid (x_{\mathrm{pa}(v)}, x_v) \neq (0, 1), \quad \text{for all} \quad v \in V\},$$

and the factorization of the joint distribution simplifies to

$$\Pr(X = x \mid \theta) = \prod_{\{v \mid x_{\mathrm{pa}(v)} = 1\}} \theta_v^{x_v} (1 - \theta_v)^{1 - x_v}.$$

The mutational pathway metaphor, originating in the virological literature, is generally considered to be a reasonable approximation to HIV evolution under drug pressure. However, sets of mutational patterns that support different tree topologies are commonly seen in clinical HIV databases. Thus, in order to allow for increased flexibility in modeling evolutionary pathways and to account for noise in the observed data, we consider the larger model class of mixtures of mutagenetic trees. Intuitively, these mixture models correspond to the assumption that a variety of evolutionary forces contribute additively in shaping HIV genetic variability *in vivo*.

Consider $K$ mutagenetic trees $T_1, \ldots, T_K$ with weights $\lambda_1, \ldots, \lambda_{K-1}$, and $\lambda_K = 1 - \sum_{k=1}^{K-1} \lambda_k$, respectively, such that $0 \leq \lambda_k \leq 1$ for all $k = 1, \ldots, K$. Each tree $T_k$ has parameters $\theta_k = (\theta_{k,v})_{v=1,\ldots,n}$. The *mutagenetic trees mixture model* is the family of distributions of $X$ of the form

$$\Pr(X = x \mid \lambda, \theta) = \sum_{k=1}^{K} \lambda_k \Pr(X = x \mid \theta_k).$$

The state space $\mathcal{C}$ of this model is the union of the state spaces of the single tree models induced by $T_1, \ldots, T_K$. In our applications, we will always fix the first tree to be a star, such that $\mathcal{C} = \{0, 1\}^n$ (i.e., all mutational patterns have non-zero probability). The star accounts for the spontaneous and independent occurrence of genetic events.

## 3   The MTreeMix Fisher kernel

We now derive a Fisher kernel for the mutagenetic trees mixture models introduced in the previous section. In this paper, our primary motivation is to improve the prediction of drug resistance

| $x, x'$ | 0 ⟋ ⟍   1   2 | 0 ↓ 1 ↓ 2 | 0 ↓ 2 ↓ 1 |
|---|---|---|---|
| 00,00 | $(\theta_1 - 1)^{-2} + (\theta_2 - 1)^{-2}$ | $(\theta_1 - 1)^{-2}$ | $(\theta_2 - 1)^{-2}$ |
| 00,01 | $(\theta_1 - 1)^{-2} + \theta_2^{-1}(\theta_2 - 1)^{-1}$ | — | $\theta_2^{-1}(\theta_2 - 1)^{-1}$ |
| 00,10 | $\theta_1^{-1}(\theta_1 - 1)^{-1} + (\theta_2 - 1)^{-2}$ | $\theta_1^{-1}(\theta_1 - 1)^{-1}$ | — |
| 00,11 | $\theta_1^{-1}(\theta_1 - 1)^{-1} + \theta_2(\theta_2 - 1)^{-1}$ | $\theta_1^{-1}(\theta_1 - 1)^{-1}$ | $\theta_2^{-1}(\theta_2 - 1)^{-1}$ |
| 01,01 | $(\theta_1 - 1)^{-2} + \theta_2^{-2}$ | — | $(\theta_1 - 1)^{-2} + \theta_2^{-2}$ |
| 01,10 | $\theta_1^{-1}(\theta_1 - 1)^{-1} + \theta_2^{-1}(\theta_2 - 1)^{-1}$ | — | — |
| 01,11 | $\theta_1^{-1}(\theta_1 - 1)^{-1} + \theta_2^{-2}$ | — | $\theta_1^{-1}(\theta_1 - 1)^{-1} + \theta_2^{-2}$ |
| 10,10 | $\theta_1^{-2} + (\theta_2 - 1)^{-2}$ | $\theta_1^{-2} + (\theta_2 - 1)^{-2}$ | — |
| 10,11 | $\theta_1^{-2} + \theta_2^{-1}(\theta_2 - 1)^{-1}$ | $\theta_1^{-2} + \theta_2^{-1}(\theta_2 - 1)^{-1}$ | — |
| 11,11 | $\theta_1^{-2}\theta_2^{-2}$ | $\theta_1^{-2}\theta_2^{-2}$ | $\theta_1^{-2}\theta_2^{-2}$ |

Table 1: Mutagenetic tree Fisher kernels for the three trees on the vertices $\{0, 1, 2\}$. The value of the kernel $K(x, x')$ is displayed for all possible pairs of mutational patterns $(x, x')$. Empty cells are indexed with genotypes that are not compatible with the tree.

from viral genotype. However, we defer application-specific details to Section 4, to emphasize the broader applicability of the kernel itself, for example in kernelized principal components analysis or multidimensional scaling.

As Jaakkola and Haussler [2] have suggested, the gradient of the log-likelihood function induced by a generative probabilistic model provides a natural comparison between samples. This is because the partial derivatives in the direction of the model parameters describe how each parameter contributes to the generation of that particular sample. Intuitively, two samples should be considered similar from this perspective, if they influence the likelihood surface in a similar way. The natural inner product for the statistical manifold induced by the log-likelihood gradient is given by the Fisher information matrix [4]. The computation of this matrix is straightforward, but for practical purposes, the Euclidean dot product $\langle \cdot, \cdot \rangle$ provides a suitable substitute for the Fisher metric [2] .

We first derive the Fisher kernel for the single mutagenetic tree model. The log-likelihood of observing a mutational pattern $x \in \{0, 1\}^n$ under this model is

$$\ell_x(\theta) = \sum_{\{v | x_{\mathrm{pa}(v)} = 1\}} x_v \log(\theta_v) + (1 - x_v) \log(1 - \theta_v).$$

Hence, the feature mapping of binary mutational patterns into Euclidean $n$-space,

$$\phi : \mathcal{C} \to \mathbb{R}^n, \quad x \mapsto \nabla \ell_x(\theta) = \left( \frac{\partial \ell_x(\theta)}{\partial \theta_1}, \dots, \frac{\partial \ell_x(\theta)}{\partial \theta_n} \right),$$

is given by the Fisher score consisting of the partial derivatives

$$\frac{\partial \ell_x(\theta)}{\partial \theta_w} = \theta_w^{-x_w}(\theta_w - 1)^{x_w - 1} 0^{1 - x_{\mathrm{pa}(w)}} = \begin{cases} \theta_w^{-1}, & \text{if } (x_{\mathrm{pa}(w)}, x_w) = (1, 1) \\ (\theta_w - 1)^{-1}, & \text{if } (x_{\mathrm{pa}(w)}, x_w) = (1, 0) \\ 0, & \text{if } (x_{\mathrm{pa}(w)}, x_w) = (0, 0). \end{cases}$$

Thus, we can define the *mutagenetic tree Fisher kernel* as

$$K(x, x') = \langle \nabla \ell_x(\theta), \nabla \ell_{x'}(\theta) \rangle = \sum_{v=1}^{n} \theta^{-(x_v + x'_v)} (\theta_v - 1)^{(x_v + x'_v) - 2} 0^{2 - (x_{\mathrm{pa}(v)} + x'_{\mathrm{pa}(v)})}.$$

For example, the Fisher kernels for the three mutagenetic trees on $n = 2$ genetic events are displayed in Table 1.

To better understand the operation of the novel kernel, we rewrite the kernel function $K$ as follows:

$$K(x, x') = \sum_{v=1}^{n} \kappa(\theta_v)_{(x_{\mathrm{pa}(v)}, x_v), (x'_{\mathrm{pa}(v)}, x'_v)},$$

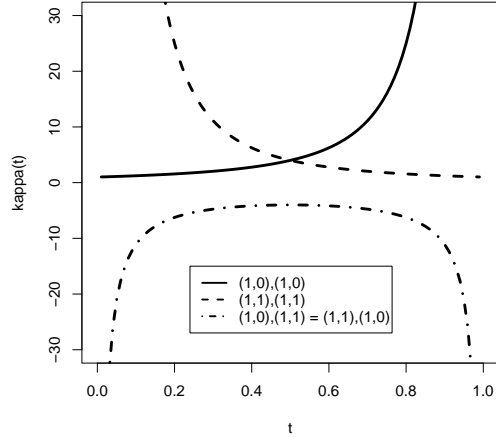

Figure 2: Non-zero entries of the matrix $\kappa(t)$ that defines the mutagenetic tree Fisher kernel. The three graphs are indexed in the same way as the matrix, namely by pairs $((x_{\mathrm{pa}(v)}, x_v), (x'_{\mathrm{pa}(v)}, x'_v))$ denoting the value of two genotypes $x$ and $x'$ at an edge $(\mathrm{pa}(v), v)$ of the mutagenetic tree. The graphs illustrate that the largest contributions stem from shared, unlikely mutations (positive effect, solid and dashed line) and from differing, likely or unlikely mutations (negative effect, dash-dot line).

with $\kappa$ defined as

$$
\kappa(t) = \begin{array}{c} \\ (0,0) \\ (1,0) \\ (1,1) \end{array} \overset{\displaystyle (0,0) \qquad (1,0) \qquad (1,1)}{\left( \begin{array}{ccc} 0 & 0 & 0 \\ 0 & (t-1)^{-2} & t^{-1}(t-1)^{-1} \\ 0 & t^{-1}(t-1)^{-1} & t^{-2} \end{array} \right)}
$$

The matrix $\kappa(t)$ is indexed by pairs of pairs $((x_{\mathrm{pa}(v)}, x_v), (x'_{\mathrm{pa}(v)}, x'_v))$. The non-zero entries of $\kappa$ are displayed in Figure 2 as functions of the parameter $t$. An edge contributes strongly to the kernel value, if the two genotypes agree on it, but the common event (occurrence or non-occurrence of the mutation) was unlikely (Figure 2, solid and dashed line). If the two genotypes disagree, the edge contributes negatively, especially for extreme parameters $\theta_v$ close to zero or one (Figure 2, dash-dot line), which make one of the events very likely and the other very unlikely. Thus, the application of the Fisher kernel idea to mutagenetic trees leads to a kernel that measures similarity of evolutionary escape in a way that corresponds well to virological intuition.

Due to the linear mixing process, extending the Fisher kernel from a single mutagenetic tree to a mixture model is straightforward. Let $\ell_x(\lambda, \theta) = \log \Pr(x \mid \lambda, \theta))$ be the log-likelihood function, and denote by

$$
\gamma_l(x \mid \lambda, \theta) = \frac{\lambda_l \Pr(x \mid \theta_l)}{\Pr(x \mid \lambda, \theta)}
$$

the responsibility of tree component $T_l$ for the observation $x$. Then the partial derivatives with respect to $\theta$ can be expressed in terms of the partials obtained for the single tree models, weighted by the responsibilities of the trees,

$$
\frac{\partial \ell_x(\lambda, \theta)}{\partial \theta_{l,w}} = \gamma_l(x \mid \lambda, \theta) \frac{\partial \ell_x(\theta_l)}{\partial \theta_{l,w}}.
$$

Differentiation with respect to $\lambda$ yields

$$
\frac{\partial \ell_x(\lambda, \theta)}{\partial \lambda_l} = \frac{\Pr(x \mid \theta_l) - \Pr(x \mid \theta_K)}{\Pr(x \mid \lambda, \theta)}.
$$

We obtain the *mutagenetic trees mixture (MTreeMix) Fisher kernel*

$$K(x, x') = \langle \nabla \ell_x(\lambda, \theta), \nabla \ell_{x'}(\lambda, \theta) \rangle$$

$$= \sum_{l=1}^{K-1} \frac{[\Pr(x \mid \theta_l) - \Pr(x \mid \theta_K)][\Pr(x' \mid \theta_l) - \Pr(x' \mid \theta_K)]}{\Pr(x \mid \lambda, \theta) \Pr(x' \mid \lambda, \theta)}$$

$$+ \sum_{l=1}^{K} \sum_{w=1}^{n} \gamma_l(x \mid \lambda, \theta) \gamma_l(x' \mid \lambda, \theta) \kappa(\theta_{l,w})_{(x_{\mathrm{pa}(w)}, x_w), (x'_{\mathrm{pa}(w)}, x'_w)}.$$

## 4  Experimental results

In this section, we use the Fisher kernel derived from mutagenetic tree mixtures for predicting HIV drug resistance from viral genotype. Briefly, resistance is the ability of a virus to replicate in the presence of drug. The degree of resistance is usually communicated as a non-negative number. This number indicates the fold-change increase in drug concentration that is necessary to inhibit viral replication by 50%, as compared to a fully susceptible reference virus. Thus, higher fold-changes correspond to increasing levels of resistance. We consider all fold-change values on a $\log_{10}$ scale.

Information on phenotypic resistance strongly affects treatment decisions, but the experimental procedures are too expensive and time-consuming for routine clinical diagnostics. Instead, at the time of therapy failure, the genotypic makeup of the viral population is determined using standard sequencing methods, leaving the challenge of inferring the phenotypic implications from the observed genotypic alterations. It is also desirable to minimize the number of sequence positions required for reliable determination of drug resistance. With a small number of positions, sequencing could be replaced by the much cheaper line-probe assay (LiPA) technology [12], which focuses on the determination of mutations at a limited number of pre-selected sites. This method could bring resistance testing to resource-poor settings in which DNA sequencing is not affordable.

All approaches to this problem described to date are based on a direct correlation between genotype and phenotype, without any further modelling involved. Application of the Fisher kernel to this task is motivated by the hypothesis that the traces of evolution present in the data and modelled by mutagenetic trees mixture models can provide additional information, leading to improved predictive performance. In a recent comparison of several statistical learning methods, support vector regression attained the highest average predictive performance across all drugs [13]. Accordingly, we have chosen this best-performing method to compare to the novel kernel.

Specifically, our experimental setup is as follows. For each drug, we start with a genotype-phenotype data set [14] of size 305 to 858 (Table 2, column 3). Based on a list of resistance mutations maintained by the International AIDS Society [15], we extract the residues listed in column 2. The number indicates the position in the viral enzyme (reverse transcriptase for the first two groups of drugs, and protease for the third group), and the amino acids following the number denote the mutations at the respective site that are considered resistance-associated. For example, the feature vector for the drug zidovudine (ZDV) consists of six variables representing the reverse transcriptase mutations 41L, 67N, 70R, 210W, 215F or Y, and 219E or Q. In the naive indicator representation, a mutational pattern within these six mutations is transformed to a binary vector of length six, each entry encoding the presence or absence of the respective mutation.

The Fisher kernel requires a mutagenetic trees mixture model for each of the evaluated drugs. Using the MTreeMix software package[1], these models were estimated from an independent set of sequences derived from patients failing a therapy that contained the specific drug of interest. In 100 replicates of ten-fold cross-validation for each drug model, we then recorded the squared correlation coefficient ($r^2$) of indicator variable-based versus Fisher kernel-based support vector regression. Avoiding both costly double cross-validation with the limited amount of data and overfitting with single cross-validation, we fixed standard parameters for both SVMs. As suggested by Jaakkola and Haussler [2], the Fisher kernel may be combined with additional transformations. Thus, we evaluated the standard kernels for both setups. For the indicator representation, the linear kernel performed best, whereas the Fisher scores performed best when combined with a Gaussian RBF kernel. We used these two kernels in the final comparison reported in Table 2.

The results displayed in columns 5 and 6 of Table 2 show the improvements attained via the Fisher kernel method as estimated by the squared correlation coefficient, $r^2$. After correction for multiple comparisons, the null hypothesis of equal mean was rejected ($P < 0.01$, Wilcoxon test) in 15 out of 17 cases, a ratio that is highly unlikely to occur by chance ($P < 0.0025$, binomial test). The most drastic improvements were obtained for the drugs 3TC, NVP and NFV. Slight decreases were observed for ddC and APV. Interestingly, when we combined both feature vectors, the cross-validated performance of the combined predictor was consistently at least as good as the best individual predictor (data not shown). We obtained similar results when evaluating performance by the mean squared error instead of the correlation coefficient (data not shown).

Table 2: Comparison of support vector regression performance for the MTreeMix Fisher kernel ($F$) versus a naive amino acid indicator ($I$) representation. The drugs (first column) are grouped into the three classes of nucleoside/nucleotide reverse transcriptase inhibitors (rows 1–7), nonnucleoside reverse transcriptase inhibitors (rows 8–10), and protease inhibitors (rows 11–17). MTreeMix models were estimated based on the mutations listed in the second column. The third column indicates the number N of available genotype-phenotype pairs, and the number K of trees in the mixture model is shown in column 4. Columns 5 and 6 indicate the squared correlation coefficients, averaged across 100 replicates of 10-fold cross-validation. P-values (last column) are obtained from Wilcoxon rank sum tests, correcting for multiple testing using the Benjamini-Hochberg method.

| DRUG | MUTATIONS | N | K | $r_F^2$ | $r_I^2$ | $\log_{10} P$ |
|---|---|---|---|---|---|---|
| ZDV | 41L, 67N, 70R, 210W, 215FY, 219EQ | 856 | 5 | 0.61 | 0.57 | $< -15.0$ |
| 3TC | 44D, 118I, 184IV | 817 | 5 | 0.71 | 0.64 | $< -15.0$ |
| ddI | 65R, 67N, 70R, 74V, 184V, 210W, 215FY, 219EQ | 858 | 4 | 0.28 | 0.24 | $< -15.0$ |
| ddC | 41L, 65R, 67N, 70R, 74V, 184V | 536 | 2 | 0.25 | 0.26 | $-0.3$ |
| d4T | 41L, 67N, 70R, 75TMSA, 210W, 215YF, 219QE | 857 | 4 | 0.22 | 0.21 | $-2.7$ |
| ABC | 41L, 65R, 67N, 70R, 74V, 115F, 184V, 210W, 215YF | 846 | 7 | 0.57 | 0.55 | $-9.0$ |
| TDF | 41L, 65R, 67N, 70R, 210W, 215YF, 219QE | 527 | 3 | 0.45 | 0.43 | $-7.0$ |
| NVP | 100I, 103N, 106A, 108I, 181CI, 188CLH, 190A | 857 | 5 | 0.58 | 0.49 | $< -15.0$ |
| EFV | 100I, 103N, 108I, 181CI, 188L, 190SA | 843 | 4 | 0.60 | 0.56 | $< -15.0$ |
| DLV | 103N, 181C | 856 | 2 | 0.49 | 0.48 | $-1.7$ |
| IDV | 10IRV, 20MR, 24I, 32I, 36I, 46IL, 54V, 71VT, 73SA, 77I, 82AFT, 84V, 90M | 851 | 4 | 0.65 | 0.63 | $-14.3$ |
| SQV | 10IRV, 48V, 54VL, 71VT, 73S, 77I, 82A, 84V, 90M | 854 | 4 | 0.68 | 0.66 | $-8.6$ |
| RTV | 10FIRV, 20MR, 24I, 32I, 33F, 36I, 46IL, 54VL, 71VT, 77I, 82AFTS, 84V, 90M | 855 | 4 | 0.77 | 0.75 | $-12.0$ |
| NFV | 10FI, 30N, 36I, 46IL, 71VT, 77I, 82AFTS, 84V, 88DS | 853 | 6 | 0.62 | 0.55 | $< -15.0$ |
| APV | 10FIRV, 32I, 46IL, 47V, 50V, 54LVM, 73S, 84V, 90M | 665 | 3 | 0.58 | 0.59 | $-2.0$ |
| LPV | 10FIRV, 20MR, 24I, 32I, 33F, 46IL, 47V, 50V, 53L, 54LV, 63P, 71VT, 73S, 82AFTS, 84V, 90M | 507 | 5 | 0.73 | 0.69 | $< -15.0$ |
| ATV | 32I, 46I, 50L, 54L, 71V, 73S, 82A, 84V, 88S, 90M | 305 | 2 | 0.54 | 0.52 | $-2.4$ |

## 5   Conclusions

The Fisher kernel derived in this paper allows for leveraging stochastic models of HIV evolution in many kernel-based scenarios. To our knowledge, this is the first study in which a probabilistic model tailored to a specific biological mechanism (namely, the evolution of drug resistance) is exploited in a discriminative context. Using the example of inferring drug resistance from viral genotype, we showed that significant improvements in predictive performance can be obtained for almost all currently available antiretroviral drugs. These results provide strong incentive for further exploitation of evolutionary models in clinical decision making. Moreover, they also underline the potential benefits from integrating several sources of data (genotype-phenotype, evolutionary). The high correlation that can be observed with a relatively small number of mutations was unexpected and suggests that reliable resistance predictions can also be obtained on the basis of LiPA assays which are much cheaper than standard sequencing technologies. While our choice of mutations was based on a selection from the literature, an interesting problem would be to design dedicated LiPA assays containing a set of mutations that allow for optimal prediction performance in this generative-discriminative setting. Finally, mixtures of mutagenetic trees have already been applied

in other contexts, for example to model progressive chromosomal alterations in cancer [16], and we expect kernel methods to play an important role in this context, too.

**Acknowledgments**

N.B. was supported by the Deutsche Forschungsgemeinschaft (BE 3217/1-1), and T.S. by the German Academic Exchange Service (D/06/41866). T.S. would like to thank Thomas Lengauer for his support and advice.

## Footnotes

[1]http://mtreemix.bioinf.mpi-sb.mpg.de

# References

[1] B. Schölkopf, K. Tsuda, and J.-P. Vert, editors. *Kernel methods in computational biology*. MIT Press, Cambridge, MA, 2004.

[2] T. Jaakkola and D. Haussler. Exploiting generative models in discriminative classifiers. In M. J. Kearns, S. A. Solla, and D. A. Cohn, editors, *Advances in Neural Information Processing Systems 11*, pages 487–493. MIT Press, Cambridge, MA, 1999.

[3] K. Tsuda, M. Kawanabe, G. Rätsch, S. Sonnenburg, and K. Müller. A new discriminative kernel from probabilistic models. In T.G. Dietterich, S. Becker, and Z. Ghahramani, editors, *Advances in Neural Information Processing Systems 14*, pages 977–984. MIT Press, Cambridge, MA, 2002.

[4] S. Amari and H. Nagaoka. *Methods of Information Geometry*. American Mathematical Society, Oxford University Press, 2000.

[5] K. Tsuda, T. Kin, and K. Asai. Marginalized kernels for biological sequences. *Bioinformatics*, 18 Suppl 1:S268–S275, 2002.

[6] J. Lafferty and G. Lebanon. Information diffusion kernels. In S. Becker, S. Thrun, and K. Obermayer, editors, *Advances in Neural Information Processing Systems 15*, pages 375–382. MIT Press, Cambridge, MA, 2003.

[7] T. Jebara, R. Kondor, and A. Howard. Probability product kernels. *Journal of Machine Learning Research*, 5:819–844, July 2004.

[8] N. Beerenwinkel, T. Sing, T. Lengauer, J. Rahnenführer, K. Roomp, I. Savenkov, R. Fischer, D. Hoffmann, J. Selbig, K. Korn, H. Walter, T. Berg, P. Braun, G. Fätkenheuer, M. Oette, J. Rockstroh, B. Kupfer, R. Kaiser, and M. Däumer. Computational methods for the design of effective therapies against drug resistant HIV strains. *Bioinformatics*, 21(21):3943–3950, Sep 2005.

[9] F. Clavel and A. J. Hance. HIV drug resistance. *N Engl J Med*, 350(10):1023–1035, Mar 2004.

[10] R. W. Shafer and J. M. Schapiro. Drug resistance and antiretroviral drug development. *J Antimicrob Chemother*, 55(6):817–820, Jun 2005.

[11] N. Beerenwinkel, J. Rahnenführer, M. Däumer, D. Hoffmann, R. Kaiser, J. Selbig, and T. Lengauer. Learning multiple evolutionary pathways from cross-sectional data. *J Comput Biol*, 12(6):584–598, 2005.

[12] J. C. Schmit, L. Ruiz, L. Stuyver, K. Van Laethem, I. Vanderlinden, T. Puig, R. Rossau, J. Desmyter, E. De Clercq, B. Clotet, and A. M. Vandamme. Comparison of the LiPA HIV-1 RT test, selective PCR and direct solid phase sequencing for the detection of HIV-1 drug resistance mutations. *J Virol Methods*, 73(1):77–82, Jul 1998.

[13] M. Rabinowitz, L. Myers, M. Banjevic, A. Chan, J. Sweetkind-Singer, J. Haberer, K. McCann, and R. Wolkowicz. Accurate prediction of HIV-1 drug response from the reverse transcriptase and protease amino acid sequences using sparse models created by convex optimization. *Bioinformatics*, 22(5):541–549, Mar 2006.

[14] H. Walter, B. Schmidt, K. Korn, A. M. Vandamme, T. Harrer, and K. Überla. Rapid, phenotypic HIV-1 drug sensitivity assay for protease and reverse transcriptase inhibitors. *J. Clin. Virol.*, 13:71–80, 1999.

[15] V. A. Johnson, F. Brun-Vezinet, B. Clotet, B. Conway, D. R. Kuritzkes, D. Pillay, J. M. Schapiro, A. Telenti, and D. D. Richman. Update of the drug resistance mutations in HIV-1: Fall 2005. *Topics in HIV Medicine*, 13(4):125–131, 2005.

[16] J. Rahnenführer, N. Beerenwinkel, W. A. Schulz, C. Hartmann, A. von Deimling, B. Wullich, and T. Lengauer. Estimating cancer survival and clinical outcome based on genetic tumor progression scores. *Bioinformatics*, 21(10):2438–2446, May 2005.
